# A Neural Model of Descending Gain Control in the Electrosensory System

Mark E. Nelson
Beckman Institute
University of Illinois
405 N. Mathews
Urbana, IL 61801

## Abstract

In the electrosensory system of weakly electric fish, descending pathways to a first-order sensory nucleus have been shown to influence the gain of its output neurons. The underlying neural mechanisms that subserve this descending gain control capability are not yet fully understood. We suggest that one possible gain control mechanism could involve the regulation of total membrane conductance of the output neurons. In this paper, a neural model based on this idea is used to demonstrate how activity levels on descending pathways could control both the gain and baseline excitation of a target neuron.

## 1 INTRODUCTION

Certain species of freshwater tropical fish, known as weakly electric fish, possess an active electric sense that allows them to detect and discriminate objects in their environment using a self-generated electric field (Bullock and Heiligenberg, 1986). They detect objects by sensing small perturbations in this electric field using an array of specialized receptors, known as electroreceptors, that cover their body surface. Weakly electric fish often live in turbid water and tend to be nocturnal. These conditions, which hinder visual perception, do not adversely affect the electric sense. Hence the electrosensory system allows these fish to navigate and capture prey in total darkness in much the same way as the sonar system of echolocating bats allows them to do the same. A fundamental difference between bat echolocation and fish

"electrolocation" is that the propagation of the electric field emitted by the fish is essentially instantaneous when considered on the time scales that characterize nervous system function. Thus rather than processing echo delays as bats do, electric fish extract information from instantaneous amplitude and phase modulations of their emitted signals.

The electric sense must cope with a wide range of stimulus intensities because the magnitude of electric field perturbations varies considerably depending on the size, distance and impedance of the object that gives rise to them (Bastian, 1981a). The range of intensities that the system experiences is also affected by the conductivity of the surrounding water, which undergoes significant seasonal variation. In the electrosensory system, there are no peripheral mechanisms to compensate for variations in stimulus intensity. Unlike the visual system, which can regulate the intensity of light arriving at photoreceptors by adjusting pupil diameter, the electrosensory system has no equivalent means for directly regulating the overall stimulus strength experienced by the electroreceptors, [1] and unlike the auditory system, there are no efferent projections to the sensory periphery to control the gain of the receptors themselves. The first opportunity for the electrosensory system to make gain adjustments occurs in a first-order sensory nucleus known as the electrosensory lateral line lobe (ELL).

In the ELL, primary afferent axons from peripheral electroreceptors terminate on the basal dendrites of a class of pyramidal cells referred to as E-cells (Maler et al., 1981; Bastian, 1981b), which represent a subset of the output neurons for the nucleus. These pyramidal cells also receive descending inputs from higher brain centers on their apical dendrites (Maler et al., 1981). One noteworthy feature is that the descending inputs are massive, far outnumbering the afferent inputs in total number of synapses. Experiments have shown that the E-cells, unlike peripheral electroreceptors, maintain a relatively constant response amplitude to electrosensory stimuli when the overall electric field normalization is experimentally altered. This automatic gain control capability is lost, however, when descending input to the ELL is blocked (Bastian, 1986ab). The underlying neural mechanisms that subserve this descending gain control capability are not yet fully understood, although it is known that GABAergic inhibition plays a role (Shumway & Maler, 1989). We suggest that one possible gain control mechanism could involve the regulation of total membrane conductance of the pyramidal cells. In the next section we present a model based on this idea and show how activity levels on descending pathways could regulate both the gain and baseline excitation of a target neuron.

## 2   NEURAL CIRCUITRY FOR DESCENDING GAIN CONTROL

Figure 1 shows a schematic diagram of neural circuitry that could provide the basis for a descending gain control mechanism. This circuitry is inspired by the circuitry found in the ELL, but has been greatly simplified to retain only the aspects that

[1]In principle, this could be achieved by regulating the strength of the fish's own electric discharge. However, these fish maintain a remarkably stable discharge amplitude and such a mechanism has never been observed.

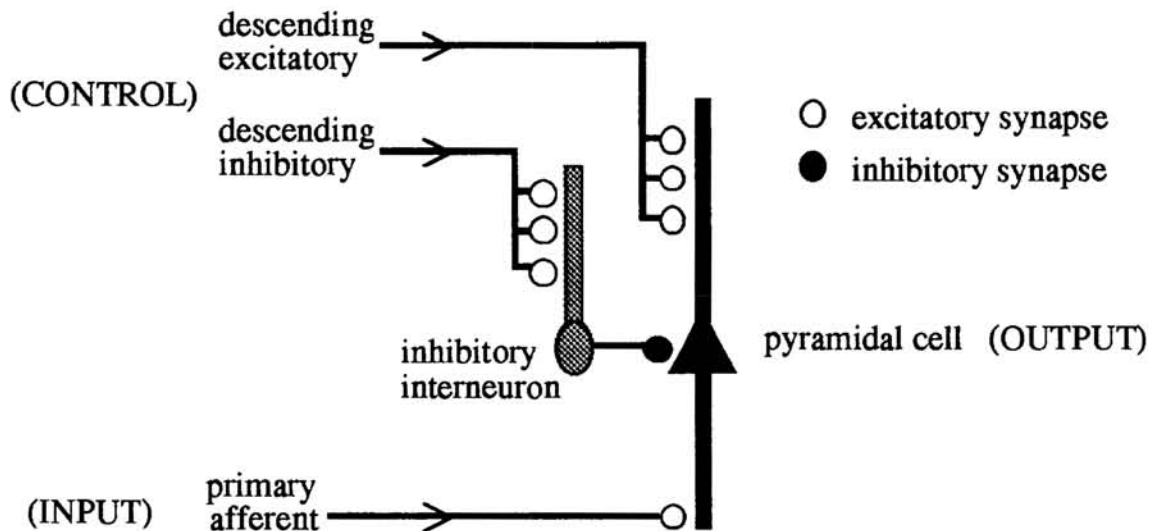

Figure 1: Neural circuitry for descending gain control. The gain of the pyramidal cell response to an input signal arriving on its basilar dendrite can be controlled by adjusting the tonic levels of activity on two descending pathways. A descending excitatory pathway makes excitatory synapses (open circles) directly on the pyramidal cell. A descending inhibitory pathway acts through an inhibitory interneuron (shown in gray) to activate inhibitory synapses (filled circles) on the pyramidal cell.

are essential for the proposed gain control mechanism. The pyramidal cell receives afferent input on a basal dendrite and control inputs from two descending pathways. One descending pathway makes excitatory synaptic connections directly on the apical dendrite of the pyramidal cell, while a second pathway exerts a net inhibitory effect on the pyramidal cell by acting through an inhibitory interneuron. We will show that under appropriate conditions, the gain of the pyramidal cell's response to an input signal arriving on its basal dendrite can be controlled by adjusting the tonic levels of activity on the two descending pathways. At this point it is worth pointing out that the spatial segregation of input and control pathways onto different parts of the dendritic tree is not an essential feature of the proposed gain control mechanism. However, by allowing independent experimental manipulation of these two pathways, this segregation has played a key role in the discovery and subsequent characterization of the gain control function in this system (Bastian, 1986ab).

The gain control function of the neural circuitry show in Figure 1 can best be understood by considering an electrical equivalent circuit for the pyramidal cell. The equivalent circuit shown in Figure 2 includes only the features that are necessary to understand the gain control function and does not reflect the true complexity of ELL pyramidal cells, which are known to contain many different types of voltage-dependent channels (Mathieson & Maler, 1988). The passive electrical properties of the circuit in Figure 2 are described by a membrane capacitance $C_m$, a leakage conductance $g_{leak}$, and an associated reversal potential $E_{leak}$. The excitatory descending pathway directly activates excitatory synapses on the pyramidal cell, giving rise to an excitatory synaptic conductance $g_{ex}$ with a reversal potential $E_{ex}$.

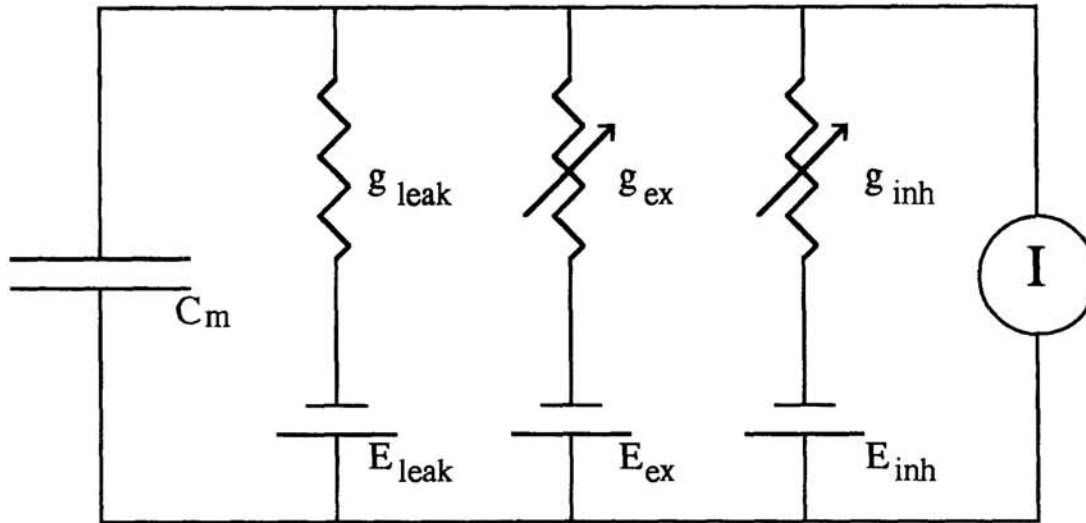

Figure 2: Electrical equivalent circuit for the pyramidal cell in the gain control circuit. The excitatory and inhibitory conductances, $g_{ex}$ and $g_{inh}$, are shown are variable resistances to indicate that their steady-state values are dependent on the activity levels of the descending pathways.

The inhibitory descending pathway acts by exciting a class of inhibitory interneurons which in turn activate inhibitory synapses on the pyramidal cell with inhibitory conductance $g_{inh}$ and reversal potential $E_{inh}$. In this model, the excitatory and inhibitory conductances $g_{ex}$ and $g_{inh}$ represent the population conductances of all the individual excitatory and inhibitory synapses associated with the descending pathways. Although individual synaptic events give rise to a *time-dependent* conductance change (which is often modeled by an $\alpha$ function), we consider the regime in which the activity levels on the descending pathways, the number of synapses involved, and the synaptic time constants are such that the summed effect can be well described by a single *time-invariant* conductance value for each pathway. The input signal (the one under the influence of the gain control mechanism) is modeled in a general form as a time-dependent current I(t). This current can represent either the synaptic current arising from activation of synapses in the primary afferent pathway, or it can represent direct current injection into the cell, such as might occur in an intracellular recording experiment.

The behavior of the membrane potential $V(t)$ for this model system is described by

$$C_m \frac{dV(t)}{dt} + g_{\ell eak}(V(t) - E_{\ell eak}) + g_{ex}(V(t) - E_{ex}) + g_{inh}(V(t) - E_{inh}) = I(t) \quad (1)$$

In the absence of an input signal ($I = 0$), the system will reach a steady-state ($dV/dt = 0$) membrane potential $V_{ss}$ given by

$$V_{ss}(I = 0) = \frac{g_{\ell eak}E_{\ell eak} + g_{ex}E_{ex} + g_{inh}E_{inh}}{g_{\ell eak} + g_{ex} + g_{inh}} \quad (2)$$

If we consider the input $I(t)$ to give rise to fluctuations in membrane potential $U(t)$ about this steady state value

$$U(t) = V(t) - V_{ss} \qquad (3)$$

then (1) can be rewritten as

$$C_m \frac{dU(t)}{dt} + g_{tot} U(t) = I(t) \qquad (4)$$

where $g_{tot}$ is the total membrane conductance

$$g_{tot} = g_{\ell eak} + g_{ex} + g_{inh} \qquad (5)$$

Equation (4) describes a first-order low-pass filter with a transfer function $G(s)$, from input current to output voltage change, given by

$$G(s) = \frac{R_{tot}}{\tau s + 1} \qquad (6)$$

where $s$ is the complex frequency ($s = i\omega$), $R_{tot}$ is the total membrane resistance ($R_{tot} = 1/g_{tot}$), and $\tau$ is the $RC$ time constant

$$\tau = R_{tot} C_m = \frac{C_m}{g_{tot}} \qquad (7)$$

The frequency dependence of the response gain $|G(i\omega)|$ is shown in Figure 3. For low frequency components of the input signal ($\omega\tau << 1$), the gain is inversely proportional to the total membrane conductance $g_{tot}$, while at high frequencies ($\omega\tau >> 1$), the gain is independent of $g_{tot}$. This is due to the fact that the impedance of the $RC$ circuit shown in Figure 2 is dominated by the resistive components at low frequencies and by the capacitive component at high frequencies. Note that the $RC$ time constant $\tau$, which characterizes the low-pass filter cutoff frequency, varies inversely with $g_{tot}$. For components of the input signal below the cutoff frequency, gain control can be accomplished by regulating the total membrane conductance.

In electrophysiological terms, this mechanism can be thought of in terms of regulating the input resistance of the neuron. As the total membrane conductance is increased, the input resistance is decreased, meaning that a fixed amount of current injection will cause a smaller change in membrane potential. Hence increasing the total membrane conductance decreases the response gain.

In our model, we propose that regulation of total membrane conductance occurs via activity on descending pathways that activate excitatory and inhibitory synaptic conductances. For this proposed mechanism to be effective, these synaptic conductances must make a significant contribution to the total membrane conductance of the pyramidal cell. Whether this condition actually holds for ELL pyramidal cells has not yet been experimentally tested. However, it is not an unreasonable assumption to make, considering recent reports that synaptic background activity can have

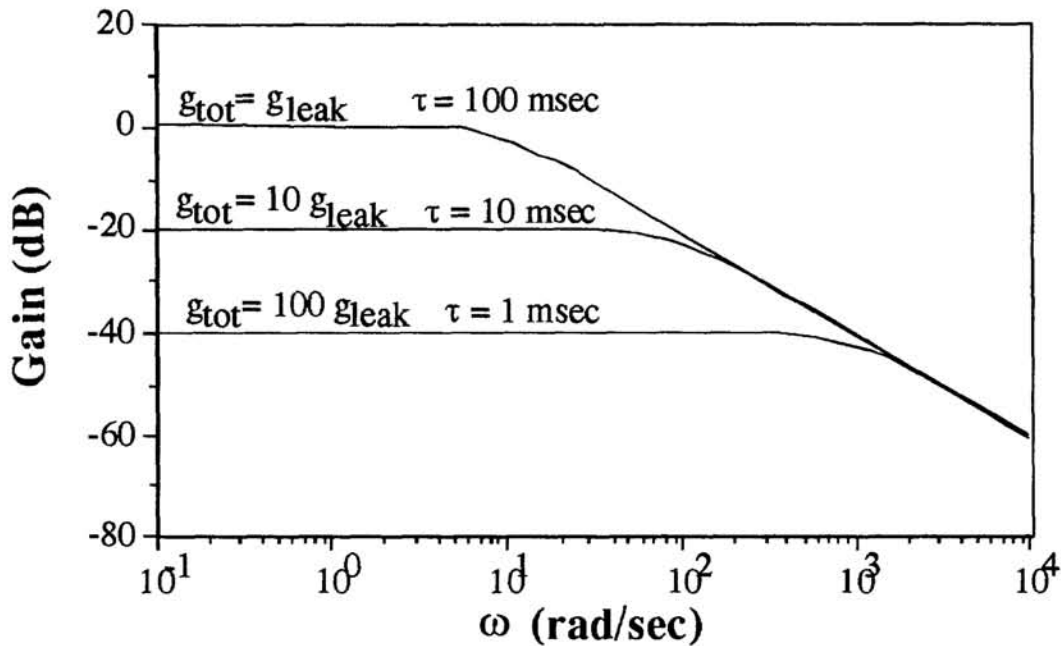

Figure 3: Gain as a function of frequency for three different values of total membrane conductance $g_{tot}$. At low frequencies, gain is inversely proportional to $g_{tot}$. Note that the time constant $\tau$, which characterizes the low-pass cutoff frequency, also varies inversely with $g_{tot}$. Gain is normalized to the maximum gain: $G_{max} = \frac{1}{g_{leak}}$; $Gain(dB) = 20\log_{10}(\frac{G}{G_{max}})$.

a significant influence on the total membrane conductance of cortical pyramidal cells (Bernander et al., 1991) and cerebellar Purkinje cells (Rapp et al., 1992).

## 3   CONTROL OF BASELINE EXCITATION

If the only functional goal was to achieve regulation of total membrane conductance, then synaptic background activity on a single descending pathway would be sufficient and there would be no need for the paired excitatory and inhibitory control pathways shown in Figure 1. However, the goal of gain control is regulate the total membrane conductance while holding the baseline level of excitation constant. In other words, we would like to be able to change the sensitivity of a neuron's response without changing its spontaneous level of activity (or steady-state resting potential if it is below spiking threshold). By having paired excitatory and inhibitory control pathways, as shown in Figure 1, we gain the extra degree-of-freedom necessary to achieve this goal.

Equation (2) provided us with a relationship between the synaptic conductances in our model and the steady-state membrane potential. In order to change the gain of a neuron, without changing its baseline level of excitation, the excitatory and inhibitory conductances must be adjusted in a way that achieves the desired total membrane conductance $g_{tot}$ and maintains a constant $V_{ss}$. Solving equations (2) and (5) simultaneously for $g_{ex}$ and $g_{inh}$, we find

$$g_{ex} = \frac{g_{tot}(V_{ss} - E_{inh}) - g_{leak}(E_{leak} - E_{inh})}{(E_{ex} - E_{inh})} \qquad (8)$$

$$g_{inh} = \frac{g_{tot}(V_{ss} - E_{ex}) - g_{leak}(E_{leak} - E_{ex})}{(E_{inh} - E_{ex})} \qquad (9)$$

For example, consider a case where the reversal potentials are $E_{leak} = -70$ mV, $E_{ex} = 0$ mV, and $E_{inh} = -90$ mV. Assume want to find values of the steady-state conductances, $g_{ex}$ and $g_{inh}$, that would result in a total membrane conductance that is twice the leakage conductance (i.e. $g_{tot} = 2g_{leak}$) and would produce a steady-state depolarization of 10 mV (i.e. $V_{ss} = -60$ mV). Using (8) and (9) we find the required synaptic conductance levels are $g_{ex} = \frac{4}{9}g_{leak}$ and $g_{inh} = \frac{5}{9}g_{leak}$.

## 4  DISCUSSION

The ability to regulate a target neuron's gain using descending control signals would provide the nervous system with a powerful means for implementing adaptive signal processing algorithms in sensory processing pathways as well as other parts of the brain. The simple gain control mechanism proposed here, involving the regulation of total membrane conductance, may find widespread use in the nervous system. Determining whether or not this is the case, of course, requires experimental verification. Even in the electrosensory system, which provided the inspiration for this model, definitive experimental tests of the proposed mechanism have yet to be carried out. Fortunately the model provides a straightforward experimentally testable prediction, namely that if activity levels on the presumed control pathways are changed, then the input resistance of the target neuron will reflect those changes. In the case of the ELL, the prediction is that if descending pathways were silenced while monitoring the input resistance of an E-type pyramidal cell, one would observe an increase in input resistance corresponding to the elimination of the descending contributions to the total membrane conductance.

We have mentioned that the gain control circuitry of Figure 1 was inspired by the neural circuitry of the ELL. For those familiar with this circuitry, it is interesting to speculate on the identity of the interneuron in the inhibitory control pathway. In the gymnotid ELL, there are at least six identified classes of inhibitory interneurons. For the proposed gain control mechanism, we are interested in the identifying those that receive descending input and which make inhibitory synapses onto pyramidal cells. Four of the six classes meet these criteria: granule cell type 2 (GC2), polymorphic, stellate, and ventral molecular layer neurons. While all four classes may participate to some extent in the gain control mechanism, one would predict, based on cell number and synapse location, that GC2 (as suggested by Shumway & Maler, 1989) and polymorphic cells would make the dominant contribution. The morphology of GC2 and polymorphic neurons differs somewhat from that shown in Figure 1. In addition to the apical dendrite, which is shown in the figure, these neurons also have a basal dendrite that receives primary afferent input. GC2 and polymorphic neurons are excited by primary afferent input and thus provide additional inhibition to pyramidal cells when afferent activity levels increase. This can be viewed as providing a feedforward component to the automatic gain control mechanism.

In this paper, we have confined our analysis to the effects of tonic changes in descending activity. While this may be a reasonable approximantion for certain experimental manipulations, it is unlikely to be a good representation of the dynamic patterns that occur under natural conditions, particularly since the descending pathways form part of a feedback loop that includes the ELL output neurons. The full story in the electrosensory system will undoubtably be much more complex. For example, there is already experimental evidence demonstrating that, in addition to gain control, descending pathways influence the spatial and temporal filtering properties of ELL output neurons (Bastian, 1986ab; Shumway & Maler, 1989).

## Acknowledgements

This work was supported by NIMH 1-R29-MH49242-01. Thanks to Joe Bastian and Lenny Maler for many enlightening discussions on descending control in the ELL.

## References

Bastian, J. (1981a) Electrolocation I: An analysis of the effects of moving objects and other electrical stimuli on the electroreceptor activity of *Apteronotus albifrons*. J. Comp. Physiol. **144**, 465-479.

Bastian, J. (1981b) Electrolocation II: The effects of moving objects and other electrical stimuli on the activities of two categories of posterior lateral line lobe cells in *Apteronotus albifrons*. J. Comp. Physiol. **144**, 481-494.

Bastian, J. (1986a) Gain control in the electrosensory system mediated by descending inputs to the electrosensory lateral line lobe. J. Neurosci. **6**, 553-562.

Bastian, J. (1986b) Gain control in the electrosensory system: a role for the descending projections to the electrosensory lateral line lobe. J. Comp. Physiol. **158**, 505-515.

Bernander, O., Douglas, R.J., Martin, K.A.C. & Koch, C. (1991) Synaptic background activity influences spatiotemporal integration in single pyramidal cells. Proc. Natl. Acad. Sci. USA **88**, 11569-11573.

Bullock, T.H. & Heiligenberg, W., eds. (1986) *Electroreception*. Wiley, New York.

Maler, L., Sas, E. and Rogers, J. (1981) The cytology of the posterior lateral line lobe of high frequency weakly electric fish (Gymnotidei): Dendritic differentiation and synaptic specificity in a simple cortex. J. Comp. Neurol. **195**, 87-140.

Mathieson, W.B. & Maler, L. (1988) Morphological and electrophysiological properties of a novel *in vitro* preparation: the electrosensory lateral line lobe brain slice. J. Comp. Physiol. **163**, 489-506.

Rapp, M., Yarom, Y. & Segev, I. (1992) The impact of parallel fiber background activity on the cable properties of cerebellar Purkinje Cells. Neural Comp. **4**, 518-533.

Shumway, C.A. & Maler, L.M. (1989) GABAergic inhibition shapes temporal and spatial response properties of pyramidal cells in the electrosensory lateral line lobe of gymnotiform fish J. Comp. Physiol. **164**, 391-407.
